# LEARNING WITH TEMPORAL DERIVATIVES IN PULSE-CODED NEURONAL SYSTEMS

Mark Gluck                David B. Parker                Eric S. Reifsnider

*Department of Psychology*
*Stanford University*
*Stanford, CA 94305*

## Abstract

A number of learning models have recently been proposed which involve calculations of temporal differences (or derivatives in continuous-time models). These models, like most adaptive network models, are formulated in terms of frequency (or activation), a useful abstraction of neuronal firing rates. To more precisely evaluate the implications of a *neuronal* model, it may be preferable to develop a model which transmits discrete pulse-coded information. We point out that many functions and properties of neuronal processing and learning may depend, in subtle ways, on the pulse-coded nature of the information coding and transmission properties of neuron systems. When compared to formulations in terms of activation, computing with temporal derivatives (or differences) as proposed by Kosko (1986), Klopf (1988), and Sutton (1988), is both more stable and easier when reformulated for a more neuronally realistic pulse-coded system. In reformulating these models in terms of pulse-coding, our motivation has been to enable us to draw further parallels and connections between real-time behavioral models of learning and biological circuit models of the substrates underlying learning and memory.

## INTRODUCTION

Learning algorithms are generally defined in terms of continuously-valued levels of input and output activity. This is true of most training methods for adaptive networks, (e.g., Parker, 1987; Rumelhart, Hinton, & Williams, 1986; Werbos, 1974; Widrow & Hoff, 1960), and also for behavioral models of animal and human learning, (e.g, Gluck & Bower, 1988a, 1988b; Rescorla & Wagner, 1972), as well as more biologically oriented models of neuronal function (e.g., Bear & Cooper, in press; Hebb, 1949; Granger, Abros-Ingerson, Staubli, & Lynch, in press; Gluck & Thompson, 1987; Gluck, Reifsnider, & Thompson, in press; McNaughton & Nadel, in press; Gluck & Rumelhart, in press). In spite of the attractive simplicity and utility of the "activation" construct

neurons use discrete trains of pulses for the transmission of information from cell to cell. Frequency (or activation) is a useful abstraction of pulse trains, especially for bridging the gap between whole-animal and single neuron behavior. To more precisely evaluate the implications of a *neuronal* model, it may be preferable to develop a model which transmits discrete pulse-coded information; it is possible that many functions and properties of neuronal processing and learning may depend, in subtle ways, on the pulse-coded nature of the information coding and transmission properties of neuron systems.

In the last few years, a number of learning models have been proposed which involve computations of temporal differences (or derivatives in continuous-time models). Klopf (1988) presented a formal real-time model of classical conditioning that predicts the magnitude of conditioned responses (CRs), given the temporal relationships between conditioned stimuli (CSs) and an unconditional stimulus (US). Klopf's model incorporates a "differential-Hebbian" learning algorithm in which *changes* in presynaptic levels of activity are correlated with *changes* in postsynaptic levels of activity. Motivated by the constraints and motives of engineering, rather than animal learning, Kosko (1986) proposed the same basic rule and provided extensive analytic insights into its properties. Sutton (1988) introduced a class of incremental learning procedures, called "temporal difference" methods, which update associative (predictive) weights according to the difference between temporally successive predictions. In addition to the applied potential of this class of algorithms, Sutton & Barto (1987) show how their model, like Klopf's (1988) model, provides a good fit to a wide range of behavioral data on classical conditioning.

These models, all of which depend on computations involving changes over time in activation levels, have been successful both for predicting a wide range of behavioral animal learning data (Klopf, 1988; Sutton & Barto, 1987) and for solving useful engineering problems in adaptive prediction (Kosko, 1986; Sutton, 1988). The possibility that these models might represent the computational properties of individual neurons, seems, at first glance, highly unlikely. However, we show by reformulating these models for pulse-coded communication (as in neuronal systems) rather than in terms of abstract activation levels, the computational soundness as well as the biological relevance of the models is improved. By avoiding the use of unstable differencing methods in computing the time-derivative of activation levels, and by increasing the error-tolerance of the computations, pulse coding will be shown to improve the accuracy and reliability of these models. The pulse coded models will also be shown to lend themselves to a closer comparison to the function of real neurons than do models that operate with activation levels. As the ability of researchers to directly measure neuronal behavior grows, the value of such close comparisons will increase. As an example, we describe here a pulse-coded version of Klopf's differential-Hebbian model of classification learning. Further details are contained in Gluck, Parker, & Reifsnider, 1988.

## Pulse-Coding in Neuronal Systems

We begin by outlining the general theory and engineering advantages of pulse-coding and then describe a pulse-coded reformulation of differential-Hebbian learning. The key idea is quite simple and can be summarized as follows: Frequency can be seen, loosely speaking, as an integral of pulses; conversely, therefore, pulses can be thought of as carrying information about the derivatives of frequency. Thus, computing with the "derivatives of frequency" is analogous to computing with pulses. As described below, our basic conclusion is that differential-Hebbian learning (Klopf, 1988; Kosko, 1986) when reformulated for a pulse-coded system is both more stable and easier to compute than is apparent when the rule is formulated in terms of frequencies. These results have important implications for any learning model which is based on computing with time-derivatives, such as Sutton's Temporal Difference model (Sutton, 1988; Sutton & Barto, 1987)

There are many ways to electrically transmit analog information from point to point. Perhaps the most obvious way is to transmit the information as a signal level. In electronic systems, for example, data that varies between 0 and 1 can be transmitted as a voltage level that varies between 0 volts and 1 volt. This method can be unreliable, however, because the receiver of the information can't tell if a constant DC voltage offset has been added to the information, or if crosstalk has occurred with a nearby signal path. To the exact degree that the signal is interfered with, the data as read by the receiver will be erroneously altered. The consequences of faults appearing in the signal are particularly serious for systems that are based on derivatives of the signal. In such systems, even a small, but sudden, unintended change in signal level can drastically alter its derivative, creating large errors.

A more reliable way to transmit analog information is to encode it as the frequency of a series of pulses. A receiver can reliably determine if it has received a pulse, even in the face of DC voltage offsets or moderate crosstalk. Most errors will not be large enough to constitute a pulse, and thus will have no effect on the transmitted information. The receiver can count the number of pulses received in a given time window to determine the frequency of the pulses. Further information on encoding analog information as the frequency of a series of pulses can be found in many electrical engineering textbooks (e.g., Horowitz & Hill, 1980).

As noted by Parker (1987), another advantage of coding an analog signal as the frequency of a series of pulses is that the time derivative of the signal can be easily and stably calculated: If $x(t)$ represents a series of pulses ($x$ equals 1 if a pulse is occuring at time $t$; otherwise it equals 0) then we can estimate the frequency, $f(t)$, of the series of pulses using an exponentially weighted time average:

$$f(t) = \mu \int_{-\infty}^{t} x(\tau) e^{-\mu(t-\tau)} d\tau$$

where μ is the decay constant. The well known formula for the derivative of $f(t)$ is

$$\frac{df(t)}{dt} = \mu \left[ x(t) - f(t) \right]$$

Thus, the time derivative of pulse-coded information can be calculated without using any unstable differencing methods, it is simply a function of presence or absence of a pulse relative to the current expectation (frequency) of pulses. As described earlier, calculation of time derivatives is a critical component of the learning algorithms proposed by Klopf (1988), Kosko (1986) and Sutton (Sutton, 1988; Sutton & Barto 1987). They are also an important aspect of 2nd order (pseudo-newtonian) extensions of the backpropogation learning rule for multi-layer adaptive "connectionist" networks (Parker, 1987).

*Summary of Klopf's Model*

Klopf (1988) proposed a model of classical conditioning which incorporates the same learning rule proposed by Kosko (1986) and which extends some of the ideas presented in Sutton and Barto's (1981) real-time generalization of Rescorla and Wagner's (1972) model of classical conditioning. The mathematical specification of Klopf's model consists of two equations: one which calculates output signals based on a weighted sum of input signals (drives) and one which determines changes in synapse efficacy due to changes in signal levels. The specification of signal output level is defined as

$$y(t) = \sum_{i=1}^{n} w_i(t) x_i(t) - \theta$$

where: $y(t)$ is the measure of postsynaptic frequency of firing at time t; $w_i(t)$ is the efficacy (positive or negative) of the $i$th synapse; $x_i(t)$ is the frequency of action potentials at the $i$th synapse; $\theta$ is the threshold of firing; and $n$ is the number of synapses on the "neuron". This equation expresses the idea that the postsynaptic firing frequency depends on the summation of the weighted presynaptic firing frequencies, $w_i(t)x_i(t)$, relative to some threshold, $\theta$. The learning mechanism is defined as

$$\Delta w_i(t) = \Delta y(t) \sum_{j=1}^{\tau} c_j \, |w_i(t-j)| \, \Delta x_i(t-j)$$

where: $\Delta w_i(t)$ is the change in efficacy of the $i$th synapse at time t; $\Delta y(t)$ is the change in postsynaptic firing at time t; $\tau$ is the longest interstimulus interval over which delayed conditioning is effective. The $c_j$ are empirically established learning rate constants -- each corresponding to a different inter-stimulus interval.

In order to accurately simulate various behavioral phenomena observed in classical conditioning, Klopf adds three ancillary assumptions to his model. First, he places a lower bound of 0 on the activation of the node. Second, he proposes that changes in synaptic

weight, $\Delta w_i(t)$, be calculated only when the change in presynaptic signal level is positive -- that is, when $\Delta x_i(t-j) > 0$. Third, he proposes separate excitatory and inhibitory weights in contrast to the single real-valued associative weights in other conditioning models (e.g., Rescorla & Wagner, 1972; Sutton & Barto, 1981). It is intriguing to note that all of these assumptions are not only sufficiently justified by constraints from behavioral data but are also motivated by neuronal constraints. For a further examination of the biological and behavioral factors supporting these assumptions see Gluck, Parker, and Reifsnider (1988).

The strength of Klopf's model as a simple formal behavioral model of classical conditioning is evident. Although the model has not yielded any new behavioral predictions, it has demonstrated an impressive ability to reproduce a wide, though not necessarily complete, range of Pavlovian behavioral phenomena with a minimum of assumptions.

Klopf (1988) specifies his learning algorithm in terms of activation or frequency levels. Because neuronal systems communicate through the transmission of discrete pulses, it is difficult to evaluate the biological plausibility of an algorithm when so formulated. For this reason, we present and evaluate a pulse-coded reformulation of Klopf's model.

*A Pulse-Coded Reformulation of Klopf's Model*

We illustrate here a pulse-coded reformulation of Klopf's (1988) model of classical conditioning. The equations that make up the model are fairly simple. A neuron is said to have fired an output pulse at time t if v(t) > e, where e is a threshold value and v(t) is defined as follows:

$$v(t) = (1-d)v(t-1) + \sum w_i(t)x_i(t) \tag{1}$$

where $v(t)$ an auxillary variable, $d$ is a small positive constant representing the leakage or decay rate, $w_i(t)$ is the efficacy of synapse $i$ at time $t$, and $x_i(t)$ is the frequency of presynaptic pulses at time $t$ at synapse $i$. The input to the decision of whether the neuron will fire consists of the weights and efficacies of the synapses as well as information about previous activation levels at the neuronal output. Note that the leakage rate, $d$, causes older information about activation levels to have less impact on current values of $v(t)$ than does recent information of the same type.

The output of the neuron, $p(t)$, is:

$v(t) > e$  then  $p(t) = 1$  (pulse generated)
$v(t) \leq e$  then  $p(t) = 0$  (no pulse generated)

It is important that once $p(t)$ has been determined, $v(t)$ will need to be adjusted if

$p(t) = 1$. To reflect the fact that the neuron has fired, (i.e., $p(t) = 1$) then $v(t) = v(t) - 1$. This decrement occurs after $p(t)$ has been determined for the current $t$. Frequencies of pulses at the output node and at the synapses are calculated using the following equations:

$$f(t) = f(t-1) + \Delta f(t)$$

where

$$\Delta f(t) = m(p(t) - f(t-1))$$

where $f(t)$ is the frequency of outgoing pulses at time $t$; $p(t)$ is the ouput (1 or 0) of the neuron at time $t$; and $m$ is a small positive constant representing a leakage rate for the frequency calculation.

Following Klopf (1988), changes in synapse efficacy occur according to

$$\Delta w_i(t) = \Delta y(t) \sum_{j=1}^{\tau} c_j \, |w_i(t-j)| \, \Delta x_i(t-j) \tag{2}$$

where

$$\Delta w_i(t) = w_i(t+1) - w_i(t)$$

and $\Delta y(t)$ and $\Delta x_i(t)$ are calculated analogously to $\Delta f(t)$; $\tau$ is the longest interstimulus interval (ISI) over which delay conditioning is effective; and $c_j$ is an empirically established set of learning rates which govern the efficacy of conditioning at an ISI of $j$. Changes in $w_i(t)$ are governed by the learning rule in Equation 2 which alters $v(t)$ via Equation 1.

Figure 1 shows the results of a computer simulation of a pulse-coded version of Klopf's conditioning model. The first graph shows the excitatory weight (dotted line) and inhibitory weight (dashed line) of the CS "synapse". Also on the same graph is the net synaptic weight (solid line), the sum of the excitatory and inhibitory weights. The subsequent graphs show CS input pulses, US input pulses, and the output (CR) pulses. The simulation consists of three acquisition trials followed by three extinction trials.

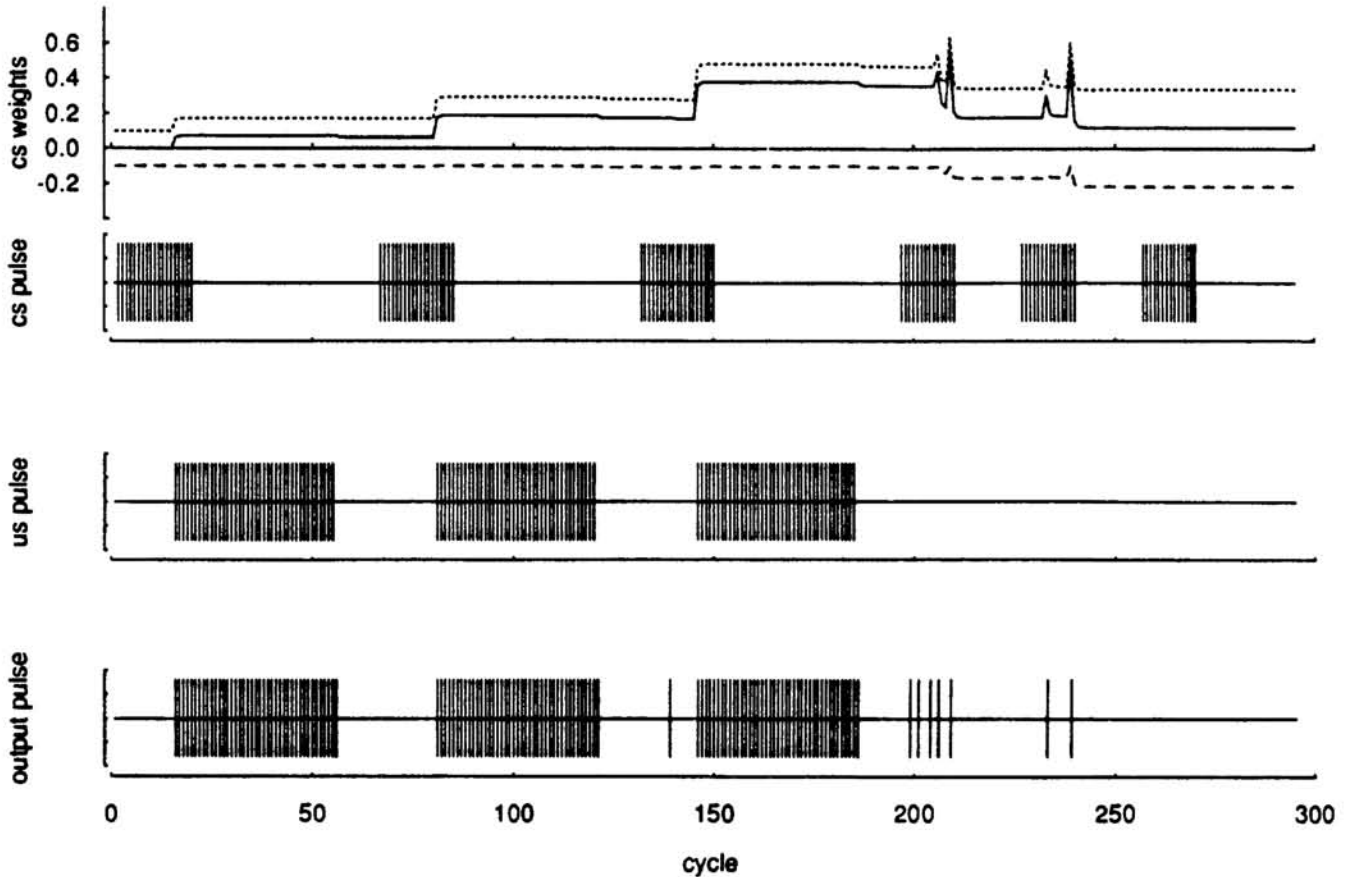

Figure 1. Simulation of pulse-coded version of Klopf's conditioning model. Top panel shows excitatory and inhibitory weights as dashed lines and the net synaptic weight of the CS as a solid line. Lower panels show the CS and US inputs and the CR output.

As expected, excitatory weight increases in magnitude over the three acquisition trials, while inhibitory weight is stable. During the first two extinction trials, the excitatory and the net synaptic weights decrease in magnitude, while the inhibitory weight increases. Thus, the CS produces a decreasing amount of output pulses (the CR). During the third extinction trial the net synaptic weight is so low that the CS cannot produce output pulses, and so the CR is extinct. However, as net weight and excitatory weight remain positive, there are residual effects of the acquisition which will accelerate reacquisition. Because a threshold must be reached before a neuronal output pulse can be emitted, and because output must occur for weight changes to occur, pulse coding adds to the accelerated reacquisition effect that is evident in the original Klopf model; extinction is halted before net weight is zero, when pulses can no longer be produced.

## Discussion

To facilitate comparison between learning algorithms involving temporal derivative computations and actual neuronal capabilities, we formulated a pulse-coded variation of Klopf's classical conditioning model. Our basic conclusion is that computing with temporal derivatives (or differences) as proposed by Kosko (1986), Klopf (1988), and Sutton (1988), is more stable and easier when reformulated for a more neuronally realistic pulse-coded system, than when the rules are formulated in terms of frequencies or activation.

It is our hope that further examination of the characteristics of pulse-coded systems may reveal facts that bear on the characteristics of neuronal function. In reformulating these algorithms in terms of pulse-coding, our motivation has been to enable us to draw further parallels and connections between real-time behavioral models of learning and biological circuit models of the substrates underlying classical conditioning. (e.g., Thompson, 1986; Gluck & Thompson, 1987; Donegan, Gluck, & Thompson, in press). More generally, noting the similarities and differences between algorithmic/behavioral theories and biological capabilities is one way of laying the groundwork for developing more complete integrated theories of the biological bases of associative learning (Donegan, Gluck, & Thompson, in press).

*Acknowledgments*

Correspondence should be addressed to: Mark A. Gluck, Dept. of Psychology, Jordan Hall; Bldg. 420, Stanford, CA 94305. For their commentary and critique on earlier drafts of this and related papers, we are indebted to Harry Klopf, Bart Kosko, Richard Sutton, and Richard Thompson. This research was supported by an Office of Naval Research Grant to R. F. Thompson and M. A. Gluck.

*References*

Bear, M. F., & Cooper, L. N. (in press). Molecular mechanisms for synaptic modification in the visual cortex: Interaction between theory and experiment. In M. A. Gluck, & D. E. Rumelhart (Eds.), *Neuroscience and Connectionist Theory*. Hillsdale, N.J.: Lawrence Erlbaum Associates..

Donegan, N. H., Gluck, M. A., & Thompson, R. F. (1989). Integrating behavioral and biological models of classical conditioning. In R. D. Hawkins, & G. H. Bower (Eds.), *Computational models of learning in simple neural systems (Volume 22 of the Psychology of Learning and Motivation)*. New York: Academic Press.

Gluck, M. A., & Bower, G. H. (1988a). Evaluating an adaptive network model of human learning. *Journal of Memory and Language, 27*, 166-195.

Gluck, M. A., & Bower, G. H. (1988b). From conditioning to category learning: An adaptive network model. *Journal of Experimental Psychology: General, 117*(3), 225-244.

Gluck, M. A., Parker, D. B., & Reifsnider, E. (1988). Some biological implications of a differential-Hebbian learning rule. *Psychobiology, 16*(3), 298-302.

Gluck, M. A., Reifsnider, E. S., & Thompson, R. F. (in press). Adaptive signal processing and temporal coarse coding: Cerebellar models of classical conditioning and VOR Adaptation. In M. A. Gluck, & D. E. Rumelhart (Eds.), *Neuroscience and Connectionist Theory*. Hillsdale, N.J.: Lawrence Erlbaum Associates..

Gluck, M. A., & Rumelhart, D. E. (in press). *Neuroscience and Connectionist Theory*. Hillsdale, N.J.: Lawrence Erlbaum Associates..

Gluck, M. A., & Thompson, R. F. (1987). Modeling the neural substrates of associative learning and memory: A computational approach. *Psychological Review, 94*, 176-191.

Granger, R., Ambros-Ingerson, J., Staubli, U., & Lynch, G. (in press). Memorial operation of multiple, interacting simulated brain structures. In M. A. Gluck, & D. E. Rumelhart (Eds.), *Neuroscience and Connectionist Theory*. Hillsdale, N.J.: Lawrence Erlbaum Associates..

Hebb, D. (1949). *Organization of Behavior*. New York: Wiley & Sons.

Horowitz, P., & Hill, W. (1980). *The Art of Electronics*. Cambridge, England: Cambridge University Press.

Klopf, A. H. (1988). A neuronal model of classical conditioning. *Psychobiology, 16*(2), 85-125.

Kosko, B. (1986). Differential hebbian learning. In J. S. Denker (Ed.), *Neural Networks for Computing, AIP Conference Proceedings 151* (pp. 265-270). New York: American Institute of Physics.

McNaughton, B. L., & Nadel, L. (in press). Hebb-Marr networks and the neurobiological representation of action in space. In M. A. Gluck, & D. E. Rumelhart (Eds.), *Neuroscience and Connectionist Theory*. Hillsdale, N.J.: Lawrence Erlbaum Associates..

Parker, D. B. (1987). Optimal Algorithms for Adaptive Networks: Second Order Back Propagation, Second Order Direct Propagation, and Second Order Hebbian Learning. *Proceedings of the IEEE First Annual Conference on Neural Networks. San Diego, California:, .*

Rescorla, R. A., & Wagner, A. R. (1972). A theory of Pavlovian conditioning: Variations in the effectiveness of reinforcement and non-reinforcement. In A. H. Black, & W. F. Prokasy (Eds.), *Classical conditioning II: Current research and theory*. New York: Appleton-Century-Crofts.

Rumelhart, D. E., Hinton, G. E., & Williams, R. J. (1986). Learning internal representations by error propogation. In D. Rumelhart, & J. McClelland (Eds.), *Parallel distributed processing: Explorations in the microstructure of cognition (Vol. 1: Foundations)*. Cambridge, M.A.: MIT Press.

Sutton, R. S. (1988). Learning to predict by the methods of temporal differences. *Machine Learning, 3*, 9-44.

Sutton, R. S., & Barto, A. G. (1981). Toward a modern theory of adaptive networks: Expectation and prediction. *Psychological Review, 88*, 135-170.

Sutton, R. S., & Barto, A. G. (1987). A temporal-difference model of classical conditioning. In *Proceedings of the 9th Annual Conference of the Cognitive Science Society*. Seattle, WA.

Thompson, R. F. (1986). The neurobiology of learning and memory. *Science, 233*, 941-947.

Werbos, P. (1974). *Beyond regression: New tools for prediction and analysis in the behavioral sciences*. Doctoral dissertation (Economics), Harvard University, Cambridge, Mass..

Widrow, B., & Hoff, M. E. (1960). Adaptive switching circuits. *Institute of Radio Engineers, Western Electronic Show and Convention, Convention Record, 4*, 96-194.
